# Model-Free Least Squares Policy Iteration

**Michail G. Lagoudakis**
Department of Computer Science
Duke University
Durham, NC 27708
*mgl@cs.duke.edu*

**Ronald Parr**
Department of Computer Science
Duke University
Durham, NC 27708
*parr@cs.duke.edu*

## Abstract

We propose a new approach to reinforcement learning which combines least squares function approximation with policy iteration. Our method is model-free and completely off policy. We are motivated by the least squares temporal difference learning algorithm (LSTD), which is known for its efficient use of sample experiences compared to pure temporal difference algorithms. LSTD is ideal for prediction problems, however it heretofore has not had a straightforward application to control problems. Moreover, approximations learned by LSTD are strongly influenced by the visitation distribution over states. Our new algorithm, Least Squares Policy Iteration (LSPI) addresses these issues. The result is an off-policy method which can use (or reuse) data collected from any source. We have tested LSPI on several problems, including a bicycle simulator in which it learns to guide the bicycle to a goal efficiently by merely observing a relatively small number of completely random trials.

## 1 Introduction

Linear least squares function approximators offer many advantages in the context of reinforcement learning. While their ability to generalize is less powerful than black box methods such as neural networks, they have their virtues: They are easy to implement and use, and their behavior is fairly transparent, both from an analysis standpoint and from a debugging and feature engineering standpoint. When linear methods fail, it is usually relatively easy to get some insight into why the failure has occurred.

Our enthusiasm for this approach is inspired by the least squares temporal difference learning algorithm (LSTD) [4]. LSTD makes efficient use of data and converges faster than other conventional temporal difference learning methods. Although it is initially appealing to attempt to use LSTD in the evaluation step of a policy iteration algorithm, this combination can be problematic. Koller and Parr [5] present an example where the combination of LSTD style function approximation and policy iteration oscillates between two bad policies in an MDP with just 4 states. This behavior is explained by the fact that linear approximation methods such as LSTD compute an approximation that is weighted by the state visitation frequencies of the policy under evaluation. Further, even if this problem is overcome, a more serious difficulty is that the state value function that LSTD learns is of no use for policy improvement when a model of the process is not available.

This paper introduces the *Least Squares Policy Iteration (LSPI)* algorithm, which extends the benefits of LSTD to control problems. First, we introduce LSQ, an algorithm that learns least squares approximations of the *state-action* ($Q$) value function, thus permitting action selection and policy improvement *without* a model. Next we introduce LSPI which uses the results of LSQ to form an approximate policy iteration algorithm. This algorithm combines the policy search efficiency of policy iteration with the data efficiency of LSTD. It is completely off policy and can, in principle, use data collected from any reasonable sampling distribution. We have evaluated this method on several problems, including a simulated bicycle control problem in which LSPI learns to guide the bicycle to the goal by observing a relatively small number of completely random trials.

## 2 Markov Decision Processes

We assume that the underlying control problem is a *Markov Decision Process* (MDP). An MDP is defined as a 4-tuple $(S, \mathcal{A}, P, R)$ where: $S$ is a finite set of states; $\mathcal{A}$ is a finite set of actions; $P$ is a *Markovian transition model* where $P(s, a, s')$ represents the probability of going from state $s$ to state $s'$ with action $a$; and $R$ is a *reward function* $R : S \times \mathcal{A} \times S \mapsto \mathbb{R}$, such that $R(s, a, s')$ represents the reward obtained when taking action $a$ in state $s$ and ending up in state $s'$.

We will be assuming that the MDP has an infinite horizon and that future rewards are discounted exponentially with a discount factor $\gamma \in [0, 1)$. (If we assume that all policies are proper, our results generalize to the undiscounted case.) A stationary policy $\pi$ for an MDP is a mapping $\pi : S \mapsto A$, where $\pi(s)$ is the action the agent takes at state $s$. The state-action value function $Q^\pi(s, a)$ is defined over all possible combinations of states and actions and indicates the expected, discounted total reward when taking action $a$ in state $s$ and following policy $\pi$ thereafter. The exact $Q$-values for all state-action pairs can be found by solving the linear system of the Bellman equations :

$$Q^\pi(s, a) = \mathcal{R}(s, a) + \gamma \sum_{s'} P(s, a, s') Q^\pi(s', \pi(s')),$$

where $\mathcal{R}(s, a) = \sum_{s'} P(s, a, s') R(s, a, s')$. In matrix format, the system becomes $Q^\pi = \mathcal{R} + \gamma \mathbf{P}^\pi Q^\pi$, where $Q^\pi$ and $\mathcal{R}$ are vectors of size $|S||A|$ and $\mathbf{P}^\pi$ is a stochastic matrix of size $(|S||A| \times |S||A|)$. $\mathbf{P}^\pi$ describes the transitions from pairs $(s, a)$ to pairs $(s', \pi(s'))$.

For every MDP, there exists an optimal policy, $\pi^*$, which maximizes the expected, discounted return of every state. *Policy iteration* is a method of discovering this policy by iterating through a sequence of monotonically improving policies. Each iteration consists of two phases. *Value determination* computes the state-action values for a policy $\pi^{(t)}$ by solving the above system. *Policy improvement* defines the next policy as $\pi^{(t+1)}(s) = \arg\max_a Q^{\pi^{(t)}}(s, a)$. These steps are repeated until convergence to an optimal policy, often in a surprisingly small number of steps.

## 3 Least Squares Approximation of Q Functions

Policy iteration relies upon the solution of a system of linear equations to find the Q values for the current policy. This is impractical for large state and action spaces. In such cases we may wish to approximate $Q^\pi$ with a parametric function approximator and do some form of approximate policy iteration. We now address the problem of finding a set of parameters that maximizes the accuracy of our approximator. A common class of approximators is the so called *linear architectures*, where the value function is approximated as a linear

weighted combination of $k$ basis functions (features):

$$\widehat{Q}^{\pi}(s, a, w) = \sum_{i=1}^{k} \phi_i(s, a)w_i = \phi(s, a)^{\mathsf{T}}w,$$

where $w$ is a set of weights (parameters). In general, $k << |S||A|$ and so, the linear system above now becomes an overconstrained system over the $k$ parameters $w$ :

$$\begin{aligned}
\mathbf{\Phi}w &\approx \mathcal{R} + \gamma \mathbf{P}^{\pi}\mathbf{\Phi}w \\
(\mathbf{\Phi} - \gamma \mathbf{P}^{\pi}\mathbf{\Phi})w &\approx \mathcal{R}
\end{aligned}$$

where $\mathbf{\Phi}$ is a $(|S||A| \times k)$ matrix. We are interested in a set of weights $w^{\pi}$ that yields a fixed point in value function space, that is a value function $Q^{\pi} = \mathbf{\Phi}w^{\pi}$ that is invariant under one step of value determination followed by orthogonal projection to the space spanned by the basis functions. Assuming that the columns of $\mathbf{\Phi}$ are linearly independent this is

$$\mathbf{\Phi}(\mathbf{\Phi}^{\mathsf{T}}\mathbf{\Phi})^{-1}\mathbf{\Phi}^{\mathsf{T}}(\mathcal{R} + \gamma\mathbf{P}^{\pi}\mathbf{\Phi}w^{\pi}) = \mathbf{\Phi}w^{\pi} \implies w^{\pi} = (\mathbf{\Phi}^{\mathsf{T}}(\mathbf{\Phi} - \gamma\mathbf{P}^{\pi}\mathbf{\Phi}))^{-1}\mathbf{\Phi}^{\mathsf{T}}\mathcal{R}.$$

We note that this is the standard fixed point approximation method for linear value functions with the exception that the problem is formulated in terms of $\mathbf{Q}$ values instead of state values. For any $\mathbf{P}^{\pi}$, the solution is guaranteed to exist for all but finitely many $\gamma$ [5].

## 4   LSQ: Learning the State-Action Value Function

In the previous section we assumed that a model $(R, \mathbf{P}^{\pi})$ of the underlying MDP is available. In many practical applications, such a model is not available and the value function or, more precisely, its parameters have to be learned from sampled data. These sampled data are tuples of the form: $(s, a, r, s')$, meaning that in state $s$, action $a$ was taken, a reward $r$ was received, and the resulting state was $s'$. These data can be collected from actual (sequential) episodes or from random queries to a generative model of the MDP. In the extreme case, they can be experiences of other agents on the same MDP. We know that the desired set of weights can be found as the solution of the system, $\mathbf{A}w^{\pi} = b$, where $\mathbf{A} = \mathbf{\Phi}^{\mathsf{T}}(\mathbf{\Phi} - \gamma\mathbf{P}^{\pi}\mathbf{\Phi})$ and $b = \mathbf{\Phi}^{\mathsf{T}}\mathcal{R}$.

The matrix $\mathbf{P}^{\pi}$ and the vector $\mathcal{R}$ are unknown and so, $\mathbf{A}$ and $b$ cannot be determined *a priori*. However, $\mathbf{A}$ and $b$ can be approximated using samples. Recall that $\mathbf{\Phi}$, $\mathbf{P}^{\pi}\mathbf{\Phi}$, and $\mathcal{R}$ are of the form:

$$\mathbf{\Phi} = \begin{pmatrix} \phi(s_1, a_1)^{\mathsf{T}} \\ \dots \\ \phi(s, a)^{\mathsf{T}} \\ \dots \\ \phi(s_{|S|}, a_{|A|})^{\mathsf{T}} \end{pmatrix} \quad \mathbf{P}^{\pi}\mathbf{\Phi} = \begin{pmatrix} \sum_{s'} P(s_1, a_1, s')\phi(s', \pi(s'))^{\mathsf{T}} \\ \dots \\ \sum_{s'} P(s, a, s')\phi(s', \pi(s'))^{\mathsf{T}} \\ \dots \\ \sum_{s'} P(s_{|S|}, a_{|A|}, s')\phi(s', \pi(s'))^{\mathsf{T}} \end{pmatrix}$$

$$\mathcal{R} = \begin{pmatrix} \sum_{s'} P(s_1, a_1, s')R(s_1, a_1, s') \\ \dots \\ \sum_{s'} P(s, a, s')R(s, a, s') \\ \dots \\ \sum_{s'} P(s_{|S|}, a_{|A|}, s')R(s_{|S|}, a_{|A|}, s') \end{pmatrix}$$

Given a set of samples, $D = \{s_{d_i}, a_{d_i}, s'_{d_i}, r_{d_i}) |\ i = 1, 2, \dots, L\}$, where the $(s_{d_i}, a_{d_i})$ are sampled from $S \times A$ according to distribution $\rho$ and the $s'_{d_i}$ are sampled according to $P(s'_{d_i}|s_{d_i}, a_{d_i})$, we can construct approximate versions of $\mathbf{\Phi}$, $\mathbf{P}^{\pi}\mathbf{\Phi}$, and $\mathcal{R}$ as follows :

$$\widehat{\mathbf{\Phi}} = \begin{pmatrix} \phi(s_{d_1}, a_{d_1})^{\mathsf{T}} \\ \dots \\ \phi(s_{d_i}, a_{d_i})^{\mathsf{T}} \\ \dots \\ \phi(s_{d_L}, a_{d_L})^{\mathsf{T}} \end{pmatrix} \qquad \widehat{\mathbf{P}^{\pi}\mathbf{\Phi}} = \begin{pmatrix} \phi(s'_{d_1}, \pi(s'_{d_1}))^{\mathsf{T}} \\ \dots \\ \phi(s'_{d_i}, \pi(s'_{d_i}))^{\mathsf{T}} \\ \dots \\ \phi(s'_{d_L}, \pi(s'_{d_L}))^{\mathsf{T}} \end{pmatrix} \qquad \widehat{\mathcal{R}} = \begin{pmatrix} r_{d_1} \\ \dots \\ r_{d_i} \\ \dots \\ r_{d_L} \end{pmatrix}$$

These approximations can be thought of as first sampling rows from $\boldsymbol{\Phi}$ according to $\rho$ and then, conditioned on these samples, as sampling terms from the summations in the corresponding rows of $\mathbf{P}^\pi\boldsymbol{\Phi}$ and $\mathcal{R}$. The sampling distribution from the summations is governed by the underlying dynamics ($P(s,a,s')$) of the process as the samples in $D$ are taken directly from the MDP.

Given $\widehat{\boldsymbol{\Phi}}$, $\widehat{\mathbf{P}^\pi\boldsymbol{\Phi}}$, and $\widehat{\mathcal{R}}$, $\mathbf{A}$ and $b$ can be approximated as

$$\widehat{\mathbf{A}} = \widehat{\boldsymbol{\Phi}}^\mathsf{T}(\widehat{\boldsymbol{\Phi}} - \gamma\widehat{\mathbf{P}^\pi\boldsymbol{\Phi}}) \qquad \text{and} \qquad \widehat{b} = \widehat{\boldsymbol{\Phi}}^\mathsf{T}\widehat{\mathcal{R}}$$

With $L$ uniformly distributed samples over pairs of states and actions $(s,a)$, the approximations $\widehat{\mathbf{A}}$ and $\widehat{b}$ are consistent approximations of the true $\mathbf{A}$ and $b$ :

$$E(\widehat{\mathbf{A}}) = \frac{L}{|S||\mathcal{A}|}\mathbf{A} \qquad \text{and} \qquad E(\widehat{b}) = \frac{L}{|S||\mathcal{A}|}b$$

The Markov property ensures that the solution $\widehat{w}^\pi$ will converge to the true solution $w^\pi$ for sufficiently large $L$ whenever $w^\pi$ exists:

$$E(\widehat{w}^\pi) = E(\widehat{\mathbf{A}}^{-1}\widehat{b}) = (\frac{L}{|S||\mathcal{A}|}\mathbf{A})^{-1}(\frac{L}{|S||\mathcal{A}|}b) = \mathbf{A}^{-1}b = w^\pi$$

In the more general case, where $\rho$ is not uniform, we will compute a weighted projection, which minimizes the $\rho$ weighted distance in the projection step. Thus, state $s$ is implicitly assigned weight $\rho(s)$ and the projection minimizes the weighted sum of squared errors with respect to $\rho$. In LSTD, for example, $\rho$ is the stationary distribution of $\mathbf{P}^\pi$, giving high weight to frequently visited states, and low weight to infrequently visited states.

As with LSTD, it is easy to see that approximations $(\widehat{\mathbf{A}}_1, \widehat{b}_1, \widehat{\mathbf{A}}_2, \widehat{b}_2)$ derived from different sets of samples $(D_1, D_2)$ can be combined additively to yield a better approximation that corresponds to the combined set of samples:

$$\widehat{\mathbf{A}} = \widehat{\mathbf{A}}_1 + \widehat{\mathbf{A}}_2 \qquad \text{and} \qquad \widehat{b} = \widehat{b}_1 + \widehat{b}_2.$$

This observation leads to an incremental update rule for $\widehat{\mathbf{A}}$ and $\widehat{b}$. Assume that initially $\widehat{\mathbf{A}} = 0$ and $\widehat{b} = 0$. For a fixed policy, a new sample $(s,a,r,s')$ contributes to the approximation according to the following update equation :

$$\widehat{\mathbf{A}} \leftarrow \widehat{\mathbf{A}} + \phi(s,a)(\phi(s,a)^\mathsf{T} - \gamma\phi(s',\pi(s'))^\mathsf{T}) \qquad \text{and} \qquad \widehat{b} \leftarrow \widehat{b} + \phi(s,a)r$$

We call this new algorithm LSQ due to its similarity to LSTD. However, unlike LSTD, it computes Q functions and does not expect the data to come from any particular Markov chain. It is a feature of this algorithm that it can use the same set of samples to compute Q values for any policy representation that offers an action choice for each $s'$ in the set. The policy merely determines which $\phi(s', \pi(s', \pi(s')))$ is added to $\widehat{\mathbf{A}}$ for each sample. Thus, LSQ can use every single sample available to it no matter what policy is under evaluation. We note that if a particular set of projection weights are desired, it is straightforward to reweight the samples as they are added to $\widehat{\mathbf{A}}$.

Notice that apart from storing the samples, LSQ requires only $O(k^2)$ space independently of the size of the state and the action space. For each sample in $D$, LSQ incurs a cost of $O(k^2)$ to update the matrices $\widehat{\mathbf{A}}$ and $\widehat{b}$ and a one time cost of $O(k^3)$ to solve the system and find the weights. Singular value decomposition (SVD) can be used for robust inversion of $\widehat{\mathbf{A}}$ as it is not always a full rank matrix.

LSQ includes LSTD as a special case where there is only one action available. It is also possible to extend LSQ to LSQ($\lambda$) in a way that closely resembles LSTD($\lambda$) [3], but in

that case the sample set must consist of complete episodes generated using the policy under evaluation, which again raises the question of bias due to sampling distribution, and prevents the reusability of samples. LSQ is also applicable in the case of infinite and continuous state and/or action spaces with no modification. States and actions are reflected only through the basis functions of the linear approximation and the resulting value function can cover the entire state-action space with the appropriate set of continuous basis functions.

## 5  LSPI: Least Squares Policy Iteration

The LSQ algorithm provides a means of learning an approximate state-action value function, $Q^\pi(s,a)$, for any fixed policy $\pi$. We now integrate LSQ into an approximate policy iteration algorithm. Clearly, LSQ is a candidate for the value determination step. The key insight is that we can achieve the policy improvement step without ever explicitly representing our policy and without any sort of model. Recall that in policy improvement, $\pi^{(t+1)}$ will pick the action $a$ that maximizes $Q^\pi(s,a)$. Since LSQ computes Q functions directly, we do not need a model to determine our improved policy; all the information we need is contained implicitly in the weights parameterizing our Q functions[1]:

$$\pi^{(t+1)}(s,w) = \arg\max_a \widehat{Q}(s,a) = \arg\max_a \phi(s,a)^\mathsf{T} w.$$

We close the loop simply by requiring that LSQ performs this maximization for each $s'$ when constructing the $\mathbf{A}$ matrix for a policy. For very large or continuous action spaces, explicit maximization over $a$ may be impractical. In such cases, some sort of global non-linear optimization may be required to determine the optimal action.

Since LSPI uses LSQ to compute approximate Q functions, it can use any data source for samples. A single set of samples may be used for the entire optimization, or additional samples may be acquired, either through trajectories or some other scheme, for each iteration of policy iteration. We summarize the LSPI algorithm in Figure 1. As with any approximate policy iteration algorithm, the convergence of LSPI is not guaranteed. Approximate policy iteration variants are typically analyzed in terms of a value function approximation error and an action selection error [2]. LSPI does not require an approximate policy representation, e.g., a policy function or "actor" architecture, removing one source of error. Moreover, the direct computation of linear Q functions from any data source, including stored data, allows the use of *all* available data to evaluate every policy, making the problem of minimizing value function approximation error more manageable.

## 6  Results

We initially tested LSPI on variants of the problematic MDP from Koller and Parr [5], essentially simple chains of varying length. LSPI easily found the optimal policy within a few iterations using actual trajectories. We also tested LSPI on the *inverted pendulum* problem, where the task is to balance a pendulum in the upright position by moving the cart to which it is attached. Using a simple set of basis functions and samples collected from random episodes (starting in the upright position and following a purely random policy), LSPI was able to find excellent policies using a few hundred such episodes [7].

Finally, we tried a bicycle balancing problem [12] in which the goal is to learn to balance and ride a bicycle to a target position located 1 km away from the starting location. Initially, the bicycle's orientation is at an angle of 90° to the goal. The state description is a six-dimensional vector $(\theta, \dot{\theta}, \omega, \dot{\omega}, \ddot{\omega}, \psi)$, where $\theta$ is the angle of the handlebar, $\omega$ is the vertical

```
LSPI (k, φ, γ, ε, π₀, D₀)

    // k : Number of basis functions
    // φ : Basis functions
    // γ : Discount factor
    // ε : Stopping criterion
    // π₀ : Initial policy, given as w₀, π₀ = π(s, w₀) (default: w₀ = 0)
    // D₀ : Initial set of samples, possibly empty

    D = D₀
    π' = π₀    // In essence, w' = w₀

    repeat
        Update D (optional)             // Add/remove samples, or leave unchanged
        π = π'                          // w = w'
        π' = LSQ (D, k, φ, γ, π)        // w' = LSQ (D, k, φ, γ, w)
    until (π ≈ π')                      // that is, (‖w − w'‖ < ε)

    return π                           // return w
```

Figure 1: The LSPI algorithm.

angle of the bicycle, and $\psi$ is the angle of the bicycle to the goal. The actions are the torque $\tau$ applied to the handlebar (discretized to $\{-2, 0, +2\}$) and the displacement of the rider $\upsilon$ (discretized to $\{-0.02, 0, +0.02\}$). In our experiments, actions are restricted to be either $\tau$ or $\upsilon$ (or nothing) giving a total of 5 actions[2]. The noise in the system is a uniformly distributed term in $[-0.02, +0.02]$ added to the displacement component of the action. The dynamics of the bicycle are based on the model described by Randløv and Alstrøm [12] and the time step of the simulation is set to 0.01 seconds.

The state-action value function $Q(s, a)$ for a fixed action $a$ is approximated by a linear combination of 20 basis functions:

$$( 1, \omega, \dot{\omega}, \omega^2, \dot{\omega}^2, \omega\dot{\omega}, \theta, \dot{\theta}, \theta^2, \dot{\theta}^2, \theta\dot{\theta}, \omega\theta, \omega\theta^2, \omega^2\theta, \psi, \psi^2, \psi\theta, \bar{\psi}, \bar{\psi}^2, \bar{\psi}\theta ),$$

where $\bar{\psi} = \pi - \psi$ for $\psi > 0$ and $\bar{\psi} = -\pi - \psi$ for $\psi < 0$. Note that the state variable $\ddot{\omega}$ is completely ignored. This block of basis functions is repeated for each of the 5 actions, giving a total of 100 basis functions and weights. Training data were collected by initializing the bicycle to a random state around the equilibrium position and running small episodes of 20 steps each using a purely random policy. LSPI was applied on training sets of different sizes and the average performance is shown in Figure 2(a). We used the same data set for each run of policy iteration and usually obtained convergence in 6 or 7 iterations. Successful policies usually reached the goal in approximately 1 km total, near optimal performance. We also show an annotated set of trajectories to demonstrate the performance improvement over multiple steps of policy iteration in Figure 2(b).

The following design decisions influenced the performance of LSPI on this problem: As is typical with this problem, we used a shaping reward [10] for the distance to the goal. In this case, we used 0.01 of the net change (in meters) in the distance to the goal. We found that when using full random trajectories, most of our sample points were not very useful; they occurred after the bicycle had already entered into a "death spiral" from which recovery was impossible. This complicated our learning efforts by biasing the samples towards hopeless parts of the space, so we decided to cut off trajectories after 20 steps. This created an additional problem because there was no terminating reward signal to indicate failure. We approximated this with an additional shaping reward, which was proportional to the

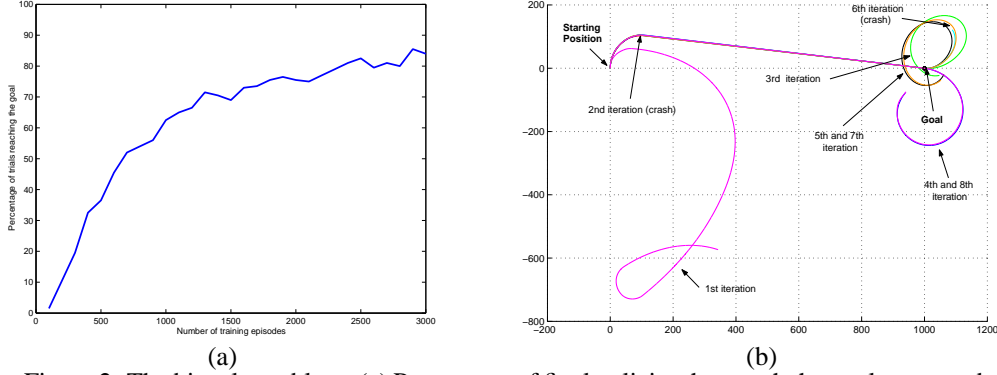

| (a) | (b) |

Figure 2: The bicycle problem: (a) Percentage of final policies that reach the goal, averaged over 200 runs of LSPI for each training set size; (b) A sample run of LSPI based on 2500 training trials. This run converged in 8 iterations. Note that iterations 5 and 7 had different Q-values but very similar policies. This was true of iterations 4 and 8 as well. The weights of the ninth differed from the eighth by less than $10^{-10}$ in $\| \cdot \|_2$, indicate convergence. The curves at the end of the trajectories indicating where the bicycle has looped back for a second pass through the goal.

net change in the square of the vertical angle. This roughly approximated the likeliness of falling at the end of a truncated trajectory. Finally, we used a discount of $0.80$, which seemed to yield more robust performance.

We admit to some slight unease about the amount of shaping and adjusting of parameters that was required to obtain good results on this problem. To verify that we had not eliminated the learning problem entirely through shaping, we reran some experiments using a discount of 0. In this case LSQ simply projects the immediate reward function into the column space of the basis functions. If the problem were tweaked too much, acting to maximize the projected immediate reward would be sufficient to obtain good performance. On the contrary, these runs always produced immediate crashes in trials.

## 7 Discussion and Conclusions

We have presented a new, model-free approximate policy iteration algorithm called LSPI, which is inspired by the LSTD algorithm. This algorithm is able to use either a stored repository of samples or samples generated dynamically from trajectories. It performs action selection and approximate policy iteration entirely in value function space, without any need for model. In contrast to other approaches to approximate policy iteration, it does not require any sort of approximate policy function.

In comparison to the memory based approach of Ormoneit and Sen [11], our method makes stronger use of function approximation. Rather than using our samples to implicitly construct an approximate model using kernels, we operate entirely in value function space and use our samples directly in the value function projection step. As noted by Boyan [3] the **A** matrix used by LSTD and LSPI can be viewed as an approximate, compressed model. This is most compelling if the columns of $\Phi$ are orthonormal. While this provides some intuitions, a proper transition function cannot be reconstructed directly from **A**, making a possible interpretation of LSPI as a model based method an area for future research.

In comparison to direct policy search methods [9, 8, 1, 13, 6], we offer the strength of policy iteration. Policy search methods typically make a large number of relatively small steps of gradient-based policy updates to a parameterized policy function. Our use of policy iteration generally results in a small number of very large steps directly in policy space.

Our experimental results demonstrate the potential of our method. We achieved good performance on the bicycle task using a very small number of randomly generated samples that were reused across multiple steps of policy iteration. Achieving this level of performance with just a linear value function architecture did require some tweaking, but the transparency of the linear architecture made the relevant issues much more salient than would be the case with any "black box" approach. We believe that the direct approach to function approximation and data reuse taken by LSPI will make the algorithm an intuitive and easy to use first choice for many reinforcement learning tasks. In future work, we plan to investigate the application of our method to multi-agent systems and the use of density estimation to control the projection weights in our function approximator.

## Acknowledgments

We would like to thank J. Randløv and P. Alstrøm for making their bicycle simulator available. We also thank C. Guestrin, D. Koller, U. Lerner and M. Littman for helpful discussions. The first author would like to thank the Lilian-Boudouri Foundation in Greece for partial financial support.

## Footnotes

[1]This is the same principle that allows action selection without a model in Q-learning. To our knowledge, this is the first application of this principle in an approximate policy iteration algorithm.

[2]Results are similar for the full 9-action case, but required more training data.

## References

[1] J. Baxter and P.Bartlett. Reinforcement learning in POMDP's via direct gradient ascent. In *Proc. 17th International Conf. on Machine Learning*, pages 41–48. Morgan Kaufmann, San Francisco, CA, 2000.

[2] D. Bertsekas and J. Tsitsiklis. *Neuro-Dynamic Programming*. Athena Scientific, Belmont, Massachusetts, 1996.

[3] Justin A. Boyan. Least-squares temporal difference learning. In I. Bratko and S. Dzeroski, editors, *Machine Learning: Proceedings of the Sixteenth International Conference*, pages 49–56. Morgan Kaufmann, San Francisco, CA, 1999.

[4] S. Bradtke and A. Barto. Linear least-squares algorithms for temporal difference learning. *Machine Learning*, 22(1/2/3):33–57, 1996.

[5] D. Koller and R. Parr. Policy iteration for factored mdps. In *Proceedings of the Sixteenth Conference on Uncertainty in Artificial Intelligence (UAI-00)*. Morgan Kaufmann, 2000.

[6] V. Konda and J. Tsitsiklis. Actor-critic algorithms. In NIPS 2000 editors, editor, *Advances in Neural Information Processing Systems 12: Proceedings of the 1999 Conference*. MIT Press, 2000.

[7] M. G. Lagoudakis and R. Parr. Model-Free Least-Squares policy iteration. Technical Report CS-2001-05, Department of Computer Science, Duke University, December 2001.

[8] A. Ng and M. Jordan. PEGASUS: A policy search method for large MDPs and POMDPs. In *Proceedings of the Sixteenth Conference on Uncertainty in Artificial Intelligence (UAI-00)*. Morgan Kaufmann, 2000.

[9] A. Ng, R. Parr, and D. Koller. Policy search via density estimation. In *Advances in Neural Information Processing Systems 12: Proceedings of the 1999 Conference*. MIT Press, 2000.

[10] Andrew Y. Ng, Daishi Harada, and Stuart Russell. Policy invariance under reward transformations: theory and application to reward shaping. In *Proc. 16th International Conf. on Machine Learning*, pages 278–287. Morgan Kaufmann, San Francisco, CA, 1999.

[11] D. Ormoneit and S. Sen. Kernel-based reinforcement learning. To appear, *Machine Learning*, 2001.

[12] J. Randløv and P. Alstrøm. Learning to drive a bicycle using reinforcement learning and shaping. In *The Fifteenth International Conference on Machine Learning*, 1998. Morgan Kaufmann.

[13] R. Sutton, D. McAllester, S. Singh, and Y. Mansour. Policy gradient methods for reinforcement learning with function approximation. In *Advances in Neural Information Processing Systems 12: Proceedings of the 1999 Conference*, 2000. MIT Press.
